# Regularized Winnow Methods

**Tong Zhang**
Mathematical Sciences Department
IBM T.J. Watson Research Center
Yorktown Heights, NY 10598
tzhang@watson.ibm.com

## Abstract

In theory, the Winnow multiplicative update has certain advantages over the Perceptron additive update when there are many irrelevant attributes. Recently, there has been much effort on enhancing the Perceptron algorithm by using regularization, leading to a class of linear classification methods called support vector machines. Similarly, it is also possible to apply the regularization idea to the Winnow algorithm, which gives methods we call regularized Winnows. We show that the resulting methods compare with the basic Winnows in a similar way that a support vector machine compares with the Perceptron. We investigate algorithmic issues and learning properties of the derived methods. Some experimental results will also be provided to illustrate different methods.

## 1 Introduction

In this paper, we consider the binary classification problem that is to determine a label $y \in \{-1, 1\}$ associated with an input vector $x$. A useful method for solving this problem is through linear discriminant functions, which consist of linear combinations of the components of the input variable. Specifically, we seek a weight vector $w$ and a threshold $\theta$ such that $w^T x < \theta$ if its label $y = -1$ and $w^T x \geq \theta$ if its label $y = 1$. Given a training set of labeled data $(x^1, y^1), \ldots, (x^n, y^n)$, a number of approaches to finding linear discriminant functions have been advanced over the years. In this paper, we are especially interested in the following two families of online algorithms: Perceptron [12] and Winnow [10]. These algorithms typically fix the threshold $\theta$ and update the weight vector $w$ by going through the training data repeatedly. They are mistake driven in the sense that the weight vector is updated only when the algorithm is not able to correctly classify an example.

For the Perceptron algorithm, the update rule is additive: if the linear discriminant function misclassifies an input training vector $x^i$ with true label $y^i$, then we update each component $j$ of the weight vector $w$ as: $w_j \leftarrow w_j + \eta x_j^i y^i$, where $\eta > 0$ is a parameter called learning rate. The initial weight vector can be taken as $w = 0$.

For the (unnormalized) Winnow algorithm (with positive weight), the update rule is multiplicative: if the linear discriminant function misclassifies an input training vector $x^i$ with true label $y^i$, then we update each component $j$ of the weight vector $w$ as: $w_j \leftarrow w_j \exp(\eta x_j^i y^i)$, where $\eta > 0$ is the learning rate parameter, and the initial weight vector can be taken as $w_j = \mu_j > 0$. The Winnow algorithm belongs to a general family of algo-

rithms called exponentiated gradient descent with unnormalized weights (EGU) [9]. There can be several variants. One is called balanced Winnow, which is equivalent to an embedding of the input space into a higher dimensional space as: $\tilde{x} = [x, -x]$. This modification allows the positive weight Winnow algorithm for the augmented input $\tilde{x}$ to have the effect of both positive and negative weights for the original input $x$. Another modification is to normalize the one-norm of the weight $w$ so that $\sum_j w_j = W$, leading to the normalized Winnow.

Theoretical properties of multiplicative update algorithms have been extensively studied since the introduction of Winnow. For linearly separable binary-classification problems, both Perceptron and Winnow are able to find a weight that separate the in-class vectors from the out-of-class vectors in the training set within a finite number of steps. However, the number of mistakes (updates) before finding a separating hyperplane can be very different [10, 9]. This difference suggests that the two algorithms serve for different purposes.

For linearly separable problems, Vapnik proposed a method that optimizes the Perceptron mistake bound which he calls "optimal hyperplane" (see [15]). The same method has also appeared in the statistical mechanical learning literature (see [1, 8, 11]), and is referred to as achieving optimal stability. For non-separable problems, a generalization of optimal hyperplane was proposed in [2] by introducing a "soft-margin" loss term. In this paper, we derive regularized Winnow methods by constructing "optimal hyperplanes" that minimize the Winnow mistake bound (rather than the Perceptron mistake bound as in an SVM). We then derive a "soft-margin" version of the algorithms for non-separable problems.

For simplicity, we shall assume $\theta = 0$ in this paper. The restriction does not cause problems in practice since one can always append a constant feature to the input data $x$, which offset the effect of $\theta$. The formulation with $\theta = 0$ can be more amenable to theoretical analysis. For an SVM, a fixed threshold also allows a simple Perceptron like numerical algorithm as described in chapter 12 of [13], and in [7]. Although more complex, a non-fixed $\theta$ does not introduce any fundamental difficulty.

The paper is organized as follows. In Section 2, we review mistake bounds for Perceptron and Winnow. Based on the bounds, we show how regularized Winnow methods can be derived by mimicking the optimal stability method (and SVM) for Perceptron. We also discuss the relationship of the newly derived methods with related methods. In Section 3, we investigate learning aspects of the newly proposed methods in a context similar to some known SVM results. An example will be given in Section 4 to illustrate these methods.

## 2   SVM and regularized Winnow

### 2.1   From Perceptron to SVM

We review the derivation of SVM from Perceptron, which serves as a reference for our derivation of regularized Winnow. Consider linearly separable problems and let $w$ be a weight that separates the in-class vectors from the out-of-class vectors in the training set. It is well known that the Perceptron algorithm computes a weight that correctly classifies all training data after at most $M$ updates (a proof can be found in [15]) where $M = \|w\|_2^2 \max_i \|x^i\|_2^2 / (\min_i w^T x^i)^2$. The weight vector $w_*$ that minimizes the right hand side of the bound is called the optimal hyperplane in [15] or the optimal stability hyperplane in [1, 8, 11]. This optimal hyperplane is the solution to the following quadratic programming problem:

$$\min_w \frac{1}{2} w^2 \qquad \text{s.t.} \quad w^T x^i y^i \geq 1 \quad \text{for } i = 1, \ldots, n.$$

For non-separable problems, we introduce a slack variable $\xi^i$ for each data point $(x^i, y^i)$ $(i = 1, \ldots, n)$, and compute a weight vector $w_*(C)$ that solves

$$\min_{w,\xi} \frac{1}{2} w^T w + C \sum_i \xi^i \quad \text{s.t.} \quad w^T x^i y^i \geq 1 - \xi^i, \quad \xi^i \geq 0 \quad \text{for } i = 1, \ldots, n.$$

Where $C > 0$ is a given parameter [15]. It is known that when $C \to \infty$, $\xi_i \to 0$ and $w_*(C)$ converges to the weight vector $w_*$ of the optimal hyperplane. We can write down the KKT condition for the above optimization problem, and let $\alpha^i$ be the Lagrangian multiplier for $w^T x^i y^i \geq 1 - \xi^i$. After elimination of $w$ and $\xi$, we obtain the following dual optimization problem of the dual variable $\alpha$ (see [15], chapter 10 for details):

$$\max_{\alpha} \sum_i \alpha^i - \frac{1}{2} (\sum_i \alpha^i x^i y^i)^2 \quad \text{s.t.} \quad \alpha^i \in [0, C] \quad \text{for } i = 1, \ldots, n.$$

The weight $w_*(C)$ is given by $w_*(C) = \sum_i \alpha^i x^i y^i$ at the optimal solution. To solve this problem, one can use the following modification of the Perceptron update algorithm (see [7] and chapter 12 of [13]): at each data point $(x^i, y^i)$, we fix all $\alpha_k$ with $k \neq i$, and update $\alpha_i$ to maximize the dual objective functional, which gives:

$$\alpha^i \to \max(\min(C, \alpha^i + \eta(1 - w^T x^i y^i)), 0),$$

where $w = \sum_i \alpha^i x^i y^i$. The learning rate $\eta$ can be set as $\eta = 1/x^{iT} x^i$ which corresponds to the exact maximization of the dual objective functional.

## 2.2  From Winnow to regularized Winnow

Similar to Perceptron, if a problem is linearly separable with a positive weight $w$, then Winnow computes a solution that correctly classifies all training data after at most $M$ updates with $M = 2W(\sum_j w_j \ln \frac{w_j \|\mu\|_1}{\mu_j \|w\|_1}) \max_i \|x^i\|_\infty^2 / \delta^2$, where $0 < \delta \leq \min_i w^T x^i y^i$, $W \geq \|w\|_1$ and the learning rate is $\eta = \delta/(W \max_i \|x^i\|_\infty^2)$. The proof of this specific bound can be found in [16] which employed techniques in [5] (also see [10] for earlier results). Note that unlike the Perceptron mistake bound, the above bound is learning rate dependent. It also depends on the prior $\mu_j > 0$ which is the initial value of $w$ in the basic Winnows.

For problems separable with positive weights, to obtain an optimal stability hyperplane associated with the Winnow mistake bound, we consider fixing $\|w\|_1$ such that $\|w\|_1 = W > 0$. It is then natural to define the optimal hyperplane as the (positive weight) solution to the following convex programming problem:

$$\min_w \sum_j w_j \ln \frac{w_j}{e\mu_j} \quad \text{s.t.} \quad w^T x^i y^i \geq 1 \quad \text{for } i = 1, \ldots, n.$$

We use $e$ to denote the base of natural logarithm. Similar to the derivation of SVM, for non-separable problems, we introduce a slack variable $\xi^i$ for each data point $(x^i, y^i)$, and compute a weight vector $w_*(C)$ that solves

$$\min_{w,\xi} \sum_j w_j \ln \frac{w_j}{e\mu_j} + C \sum_i \xi^i \quad \text{s.t.} \quad w^T x^i y^i \geq 1 - \xi^i, \quad \xi^i \geq 0 \quad \text{for } i = 1, \ldots, n.$$

Where $C > 0$ is a given parameter. Note that to derive the above methods, we have assumed that $\|w\|_1$ is fixed at $\|w\|_1 = \|\mu\|_1 = W$, where $W$ is a given parameter. This implies that the derived methods are in fact regularized versions of the normalized Winnow. One can also ignore this normalization constraint so that the derived methods correspond to regularized versions of the unnormalized Winnow. The entropy regularization condition is natural

to all exponentiated gradient methods [9], as can be observed from the theoretical results in [9]. The regularized normalized Winnow is closely related to the maximum entropy discrimination [6] (the two methods are almost identical for linearly separable problems). However, in the framework of maximum entropy discrimination, the Winnow connection is non-obvious. As we shall show later, it is possible to derive interesting learning bounds for our methods that are connected with the Winnow mistake bound.

Similar to the SVM formulation, the non-separable formulation of regularized Winnow approaches the separable formulation as $C \to \infty$. We shall thus only focus on the non-separable case below. Also similar to an SVM, we can write down the KKT condition and let $\alpha^i$ be the Lagrangian multiplier for $w^T x^i y^i \geq 1 - \xi^i$. After elimination of $w$ and $\xi$, we obtain (the algebra resembles that of [15], chapter 10, which we shall skip due to the limitation of space) the following dual formulation for regularized unnormalized Winnow:

$$\max_\alpha \sum_i \alpha^i - \sum_j \mu_j \exp(\sum_i \alpha^i x_j^i y^i) \qquad \text{s.t.} \quad \alpha^i \in [0, C] \quad \text{for } i = 1, \ldots, n.$$

The $j$-th component of weight $w_*(C)$ is given by $w_*(C)_j = \mu_j \exp(\sum_i \alpha^i x_j^i y^i)$ at the optimal solution. For regularized normalized Winnow with $\|w\|_1 = W > 0$, we obtain

$$\max_\alpha \sum_i \alpha^i - W \ln(\sum_j \mu_j \exp(\sum_i \alpha^i x_j^i y^i)) \qquad \text{s.t.} \quad \alpha^i \in [0, C] \quad \text{for } i = 1, \ldots, n.$$

The weight $w_*(C)$ is given by $w_*(C)_j = W \mu_j \exp(\sum_i \alpha^i x_j^i y^i) / \sum_j \mu_j \exp(\sum_i \alpha^i x_j^i y^i)$ at the optimal solution.

Similar to the Perceptron-like update rule for the dual SVM formulation, it is possible to derive Winnow-like update rules for the regularized Winnow formulations. At each data point $(x^i, y^i)$, we fix all $\alpha_k$ with $k \neq i$, and update $\alpha_i$ to maximize the dual objective functionals. We shall not try to derive an analytical solution, but rather use a gradient ascent method with a learning rate $\eta$: $\alpha_i \to \alpha_i + \eta \frac{\partial}{\partial \alpha_i} L_D(\alpha_i)$, where we use $L_D$ to denote the dual objective function to be maximized. $\eta$ can be either fixed as a small number or computed by the Newton's method. It is not hard to verify that we obtain the following update rule for regularized unnormalized Winnow:

$$\alpha^i \to \max(\min(C, \alpha^i + \eta(1 - w^T x^i y^i)), 0),$$

where $w_j = \mu_j \exp(\sum_i \alpha^i x_j^i y^i)$. This gradient ascent on the dual variable gives an EGU rule as in [9]. Compared with the SVM dual update rule which is a soft-margin version of the Perceptron update rule, this method naturally corresponds to a soft-margin version of unnormalized Winnow update. Similarly, we obtain the following dual update rule for regularized normalized Winnow:

$$\alpha^i \to \max(\min(C, \alpha^i + \eta(1 - w^T x^i y^i)), 0),$$

where $w_j = W \mu_j \exp(\sum_i \alpha^i x_j^i y^i) / \sum_j \mu_j \exp(\sum_i \alpha^i x_j^i y^i)$. Again, this rule (which is an EG rule in [9]) can be naturally regarded as the soft-margin version of the normalized Winnow update. In our experience, these update rules are numerically very efficient. Note that for regularized normalized Winnow, the normalization constant $W$ needs to be carefully chosen based on the data. For example, if data is infinity-norm bounded by 1, then it does not seem to be appropriate if we choose $W \leq 1$ since $|w^T x| \leq 1$: a hyperplane with $\|w\|_1 \leq 1$ does not achieve reasonable margin. This problem is less crucial for unnormalized Winnow, but the norm of the initial weight $\mu_j$ still affects the solution.

Besides maximum entropy discrimination which is closely related to regularized normalized Winnow, a large margin version of unnormalized Winnow has also been proposed based on some heuristics [3, 4]. However, their algorithm was purely mistake driven without dual variables $\alpha^i$ (the algorithm does not compute an optimal stability hyperplane for the Winnow mistake bound). In addition, they did not include a regularization parameter $C$ which in practice may be important for non-separable problems.

# 3   Some statistical properties of regularized Winnows

In this section, we derive some learning bounds based on our formulations that minimize the Winnow mistake bound. The following result is an analogy of a leave-one-out cross-validation bound for separable SVMs — Theorem 10.7 in [15].

**Theorem 3.1** *The expected misclassification error* $\mathrm{err}_n$ *with the true distribution by using hyperplane $w$ obtained from the linearly separable ($C = \infty$) unnormalized regularized Winnow algorithm with $n$ training samples is bounded by* $\mathrm{err}_n \leq \frac{1}{n+1} E \min(K, 1.5 W(\sum_j w_j \ln \frac{w_j}{\mu_j}) \max_i \|x^i\|_\infty^2)$, *where the right-hand side expectation is taken with $n+1$ random samples $(x^1, y^1), \ldots, (x^{n+1}, y^{n+1})$. $K$ is the number of support vectors of the solution. Let $w$ be the optimal solution using all the samples with dual $\alpha^i$ for $i = 1, \ldots, n+1$. Let $w^k$ be the weight obtained from setting $\alpha^k = 0$, then $W = \max(\|w\|_1, \|w^1\|_1, \ldots, \|w^{n+1}\|_1)$.*

*Proof Sketch.* We only describe the major steps due to the limitation of space. Denote by $\tilde{w}^k$ the weight obtained from the optimal solution by removing $(x^k, y^k)$ from the training sample. Similar to the proof of Theorem 10.7 in [15], we need to bound the leave-one-out cross-validation error, which is at most $K$. Also note that the leave-one-out cross-validation error is at most $|\{k : \|\tilde{w}^k - w\|_1 \|x^k\|_\infty \geq 1\}|$. We then use the following two inequalities: $\|\tilde{w}^k - w\|_1^2 \leq 2W(\sum_j \tilde{w}_j^k - w_j - w_j \ln(\tilde{w}_j^k / w_j))$; and $\sum_j \tilde{w}_j^k - w_j - w_j \ln(\tilde{w}_j^k / w_j) \leq \sum_j w_j^k - w_j - w_j \ln(w_j^k / w_j)$ — the latter inequality can be obtained by comparing the dual objective functionals and by using the corresponding KKT condition of the dual problem. The remaining problem is now reduced to proving that $|\{k : \sum_j w_j^k - w_j - w_j \ln(w_j^k / w_j) \geq 1/(2W\|x^k\|_\infty^2)\}| \leq \sqrt{2}W \sum_j w_j \ln \frac{w_j}{\mu_j}$. For the dual formulation, by summing over index $k$ of the KKT first order condition with respective to the dual $\alpha^k$, multiplied by $\alpha^k$, one obtains $\sum_k \alpha^k = \sum_j w_j \ln \frac{w_j}{\mu_j}$. We thus only need to show that if $\sum_j w_j^k - w_j - w_j \ln(w_j^k / w_j) \geq 1/(2W\|x^k\|_\infty^2)$, then $\alpha^k \geq 2/(3W\|x^k\|_\infty^2)$. This can be checked directly through Taylor expansion. $\square$

By using the same technique, we may also obtain a bound for regularized normalized Winnow. One disadvantage of the above bound is that it is the expectation of a random estimator that is no better than the leave-one-out cross-validation error based on observed data. However, the bound does convey some useful information: for example, we can observe that the expected misclassification error (learning curve) converges at a rate of $O(1/n)$ as long as $W(\sum_j w_j \ln \frac{w_j}{\mu_j})$ and $\sup \|x\|_\infty$ are reasonably bounded.

It is also not difficult to obtain interesting PAC style bounds by using the covering number result for entropy regularization in [16] and ideas in [14]. Although the PAC analysis would imply a slightly suboptimal learning curve of $O(\log n / n)$ for linearly separable problems, the bound itself provides a probability confidence and can be generalized to non-separable problems. We state below an example for non-separable problems, which justifies the entropy regularization. The bound itself is a direct consequence of Theorem 2.2 and a covering number result with entropy regularization in [16]. Note that as in [14], the square root can be removed if $k_\gamma = 0$; $\gamma$ can also be made data-dependent.

**Theorem 3.2** *If the data is infinity-norm bounded as $\|x\|_\infty \leq b$, then consider the family $\Gamma$ of hyperplanes $w$ such that $\|w\|_1 \leq a$ and $\sum_j w_j \ln(\frac{w_j \|\mu\|_1}{\mu_j \|w\|_1}) \leq c$. Denote by $\mathrm{err}(w)$ the misclassification error of $w$ with the true distribution. Then there is a constant $C$ such that for any $\gamma > 0$, with probability $1 - \eta$ over $n$ random samples, any $w \in \Gamma$ satisfies:*

$$\mathrm{err}(w) \leq \frac{k_\gamma}{n} + \sqrt{\frac{C}{\gamma^2 n} b^2 (a^2 + ac) \ln(\frac{nab}{\gamma} + 2) + \ln \frac{1}{\eta}},$$

*where $k_\gamma = |\{i : w^T x^i y^i < \gamma\}|$ is the number of samples with margin less than $\gamma$.*

## 4  An example

We use an artificial dataset to show that a regularized Winnow can enhance a Winnow just like an SVM can enhance a Perceptron. In addition, it shows that for problems with many irrelevant features, the Winnow algorithms are superior to the Perceptron family algorithms.

The data in this experiment are generated as follows. We select an input data dimension $d$, with $d = 500$ or $d = 5000$. The first 5 components of the target linear weight $w$ are set to ones; the 6th component is -1; and the remaining components are zeros. The linear threshold $\theta$ is 2. Data are generated as random vectors with each component randomly chosen to be either 0 or 1 with probability 0.5 each. Five percent of the data are given wrong labels. The remaining data are given correct labels, but we remove data with margins that are less than 1. One thousand training and one thousand test data are generated.

We shall only consider balanced versions of the Winnows. We also compensate the effect of $\theta$ by appending a constant 1 to each data point, as mentioned earlier. We use UWin and NWin to denote the basic unnormalized and normalized Winnows respectively. LM-UWin and LM-NWin denote the corresponding large margin versions. The SVM style large margin Perceptron is denoted as LM-Perc. We use 200 iterations over the training data for all algorithms. The initial values for the Winnows are set to be the priors: $\mu_j = 0.01$. For online algorithms, we fix the learning rates at 0.01. For large margin Winnows, we use learning rates $\eta = 0.01$ in the gradient ascent update. For (2-norm regularized) large margin Perceptron, we use the exact update which corresponds to a choice $\eta = 1/x^{iT}x^i$.

Accuracies (in percentage) of different methods are listed in Table 1. For regularization methods, accuracies are reported with the optimal regularization parameters. The superiority of the regularized Winnows is obvious, especially for high dimensional data. Accuracies of regularized algorithms with different regularization parameters are plotted in Figure 1. These behaviors are very typical for regularized algorithms. In practice, the optimal regularization parameter can be found by cross-validation.

| dimension | Perceptron | LM-Perc | UWin | LM-UWin | NWin | LM-NWin |
|-----------|-----------|---------|------|---------|------|---------|
| 500       | 82.2      | 87.1    | 82.4 | 94.0    | 82.4 | 94.3    |
| 5000      | 67.9      | 69.8    | 69.7 | 87.4    | 69.7 | 88.6    |

Table 1: Testset accuracy (in percentage) on the artificial dataset

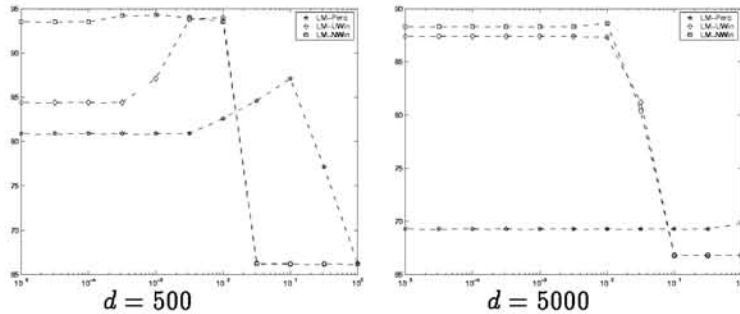

$d = 500$          $d = 5000$

Figure 1: Testset accuracy (in percentage) as a function of $\lambda = \frac{1}{nC}$

## 5 Conclusion

In this paper, we derived regularized versions of Winnow online update algorithms. We studied algorithmic and theoretical properties of the newly obtained algorithms, and compared them to the Perceptron family algorithms. Experimental results indicated that for problems with many irrelevant features, the Winnow family algorithms are superior to Perceptron family algorithms. This is consistent with the implications from both the online learning theory, and learning bounds obtained in this paper.

## References

[1] J.K. Anlauf and M. Biehl. The AdaTron: an adaptive perceptron algorithm. *Europhys. Lett.*, 10(7):687–692, 1989.

[2] C. Cortes and V.N. Vapnik. Support vector networks. *Machine Learning*, 20:273–297, 1995.

[3] I. Dagan, Y. Karov, and D. Roth. Mistake-driven learning in text categorization. In *Proceedings of the Second Conference on Empirical Methods in NLP*, 1997.

[4] A. Grove and D. Roth. Linear concepts and hidden variables. *Machine Learning*, 2000. To Appear; early version appeared in NIPS-10.

[5] A.J. Grove, N. Littlestone, and D. Schuurmans. General convergence results for linear discriminant updates. In *Proc. 10th Annu. Conf. on Comput. Learning Theory*, pages 171–183, 1997.

[6] Tommi Jaakkola, Marina Meila, and Tony Jebara. Maximum entropy discrimination. In S.A. Solla, T.K. Leen, and K.-R. Müller, editors, *Advances in Neural Information Processing Systems 12*, pages 470–476. MIT Press, 2000.

[7] T.S. Jaakkola, Mark Diekhans, and D. Haussler. A discriminative framework for detecting remote protein homologies. *Journal of Computational Biology*, to appear.

[8] W. Kinzel. Statistical mechanics of the perceptron with maximal stability. In *Lecture Notes in Physics*, volume 368, pages 175–188. Springer-Verlag, 1990.

[9] J. Kivinen and M.K. Warmuth. Additive versus exponentiated gradient updates for linear prediction. *Journal of Information and Computation*, 132:1–64, 1997.

[10] N. Littlestone. Learning quickly when irrelevant attributes abound: a new linear-threshold algorithm. *Machine Learning*, 2:285–318, 1988.

[11] M. Opper. Learning times of neural networks: Exact solution for a perceptron algorithm. *Phys. Rev. A*, 38(7):3824–3826, 1988.

[12] F. Rosenblatt. *Principles of Neurodynamics: Perceptrons and the Theory of Brain Mechanisms*. Spartan, New York, 1962.

[13] Bernhard Schölkopf, Christopher J. C. Burges, and Alexander J. Smola, editors. *Advances in Kernel Methods : Support Vector Learning*. The MIT press, 1999.

[14] J. Shawe-Taylor, P.L. Bartlett, R.C. Williamson, and M. Anthony. Structural risk minimization over data-dependent hierarchies. *IEEE Trans. Inf. Theory*, 44(5):1926–1940, 1998.

[15] V.N. Vapnik. *Statistical learning theory*. John Wiley & Sons, New York, 1998.

[16] Tong Zhang. Analysis of regularized linear functions for classification problems. Technical Report RC-21572, IBM, 1999. Abstract in NIPS'99, pp. 370–376.
